# On Triangular versus Edge Representations — Towards Scalable Modeling of Networks

**Qirong Ho**
School of Computer Science
Carnegie Mellon University
Pittsburgh, PA 15213
qho@cs.cmu.edu

**Junming Yin**
School of Computer Science
Carnegie Mellon University
Pittsburgh, PA 15213
junmingy@cs.cmu.edu

**Eric P. Xing**
School of Computer Science
Carnegie Mellon University
Pittsburgh, PA 15213
epxing@cs.cmu.edu

## Abstract

In this paper, we argue for representing networks as a bag of *triangular motifs*, particularly for important network problems that current model-based approaches handle poorly due to computational bottlenecks incurred by using edge representations. Such approaches require both 1-edges and 0-edges (missing edges) to be provided as input, and as a consequence, approximate inference algorithms for these models usually require $\Omega(N^2)$ time per iteration, precluding their application to larger real-world networks. In contrast, triangular modeling requires less computation, while providing equivalent or better inference quality. A triangular motif is a vertex triple containing 2 or 3 edges, and the number of such motifs is $\Theta(\sum_i D_i^2)$ (where $D_i$ is the degree of vertex $i$), which is much smaller than $N^2$ for low-maximum-degree networks. Using this representation, we develop a novel mixed-membership network model and approximate inference algorithm suitable for large networks with low max-degree. For networks with high maximum degree, the triangular motifs can be naturally subsampled in a *node-centric* fashion, allowing for much faster inference at a small cost in accuracy. Empirically, we demonstrate that our approach, when compared to that of an edge-based model, has faster runtime and improved accuracy for mixed-membership community detection. We conclude with a large-scale demonstration on an $N \approx 280,000$-node network, which is infeasible for network models with $\Omega(N^2)$ inference cost.

## 1 Introduction

Network analysis methods such as MMSB [1], ERGMs [20], spectral clustering [17] and latent feature models [12] require the adjacency matrix $A$ of the network as input, reflecting the natural assumption that networks are best represented as a *set of edges* taking on the values 0 (absent) or 1 (present). This assumption is intuitive, reasonable, and often necessary for some tasks, such as *link prediction*, but it comes at a cost (which is not always necessary, as we will discuss later) for other tasks, such as *community detection* in both the single-membership or admixture (mixed-membership) settings. The fundamental difference between link prediction and community detection is that the first is concerned with link outcomes on *pairs* of vertices, for which providing links as input is intuitive. However, the second task is about discovering the community memberships of *individual* vertices, and links are in fact no longer the only sensible representation. By representing the input network as a *bag of triangular motifs* — by which we mean vertex triples with 2 or 3 edges — one can design novel models for mixed-membership community detection that outperform models based on the adjacency matrix representation.

The main advantage of the bag-of-triangles representation lies in its huge reduction of computational cost for certain network analysis problems, with little or no loss of outcome quality. In the traditional edge representation, if $N$ is the number of vertices, then the adjacency matrix has size $\Theta(N^2)$ — thus, any network analysis algorithm that touches every element must have $\Omega(N^2)$ runtime complexity. For probabilistic network models, this statement applies to the cost of approximate

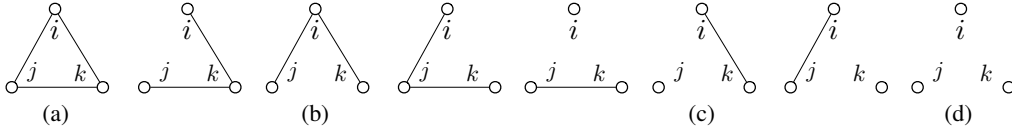

Figure 1: Four types of triangular motifs: (a) full-triangle; (b) 2-triangle; (c) 1-triangle; (d) empty-triangle. For mixed-membership community detection, we only focus on full-triangles and 2-triangles.

inference. For example, the Mixed Membership Stochastic Blockmodel (MMSB) [1] has $\Theta(N^2)$ latent variables, implying an inference cost of $\Omega(N^2)$ per iteration. Looking beyond, the popular $p^*$ or Exponential Random Graph models [20] are normally estimated via MCMC-MLE, which entails drawing network samples (each of size $\Theta(N^2)$) from some importance distribution. Finally, latent factor models such as [12] only have $\Theta(N)$ latent variables, but the Markov blanket of each variable depends on $\Theta(N)$ observed variables, resulting in $\Omega(N^2)$ computation per sweep over all variables. With an inference cost of $\Omega(N^2)$, even modestly large networks with only $\sim 10,000$ vertices are infeasible, to say nothing of modern social networks with millions of vertices or more.

On the other hand, it can be shown that the number of 2- and 3-edge triangular motifs is upper-bounded by $\Theta(\sum_i D_i^2)$, where $D_i$ is the degree of vertex $i$. For networks with low maximum degree, this quantity is $\ll N^2$, allowing us to construct more parsimonious models with faster inference algorithms. Moreover, for networks with high maximum degree, one can subsample $\Theta(N\delta^2)$ of these triangular motifs in a *node-centric* fashion, where $\delta$ is a user-chosen parameter. Specifically, we assign triangular motifs to nodes in a natural manner, and then subsample motifs only from nodes with too many of them. In contrast, MMSB and latent factor models rely on distributions over 0/1-edges (i.e. edge probabilities), and for real-world networks, these distributions cannot be preserved with small (i.e. $o(N^2)$) sample sizes because the 0-edges asymptotically outnumber the 1-edges.

As we will show, a triangular representation does not preserve *all* information found in an edge representation. Nevertheless, we argue that one should represent complex data objects in a task-dependent manner, especially since computational cost is becoming a bottleneck for real-world problems like analyzing web-scale network data. The idea of transforming the input representation (e.g. from network to bag-of-triangles) for better task-specific performance is not new. A classic example is the *bag-of-words* representation of a document, in which the ordering of words is discarded. This representation has proven effective in natural language processing tasks such as topic modeling [2], even though it eliminates practically all grammatical information. Another example from computer vision is the use of *superpixels* to represent images [3, 4]. By grouping adjacent pixels into larger superpixels, one obtains a more compact image representation, in turn leading to faster and more meaningful algorithms. When it comes to networks, triangular motifs (Figure 1) are already of significant interest in biology [13], social science [19, 9, 10, 16], and data mining [21, 18, 8]. In particular, 2- and 3-edge triangular motifs are central to the notion of transitivity in the social sciences — if we observe edges A-B and B-C, does A have an edge to C as well? Transitivity is of special importance, because high transitivity (i.e. we frequently observe the third edge A-C) intuitively leads to stronger clusters with more within-cluster edges. In fact, the ratio of 3-edge triangles to connected vertex triples (i.e. 2- and 3-edge triangular motifs) is precisely the definition of the network clustering coefficient [16], which is a popular measure of cluster strength.

In the following sections, we begin by characterizing the triangular motifs, following which we develop a mixed-membership model and inference algorithm based on these motifs. Our model, which we call MMTM or the Mixed-Membership Triangular Model, performs mixed-membership community detection, assigning each vertex $i$ to a mixture of communities. This allows for better outlier detection and more informative visualization compared to single-membership modeling. In addition, mixed-membership modeling has two key advantages: first, MM models such as MMSB, Latent Dirichlet Allocation and our MMTM are easily modified for specialized tasks — as evidenced by the rich literature on topic models [2, 1, 14, 5]. Second, MM models over disparate data types (text, network, etc.) can be combined by fusing their latent spaces, resulting in a *multi-view model* — for example, [14, 5] model both text and network data from the same mixed-membership vectors. Thus, our MMTM can serve as a basic modeling component for massive real-world networks with copious side information. After developing our model and inference algorithm, we present simulated experiments comparing them on a variety of network types to an adjacency-matrix-based model (MMSB) and its inference algorithm. These experiments will show that triangular mixed-membership modeling results in both faster inference and more accurate mixed-membership recovery. We conclude by demonstrating our model/algorithm on a network with $N \approx 280,000$ nodes and $\sim 2,300,000$ edges, which is far too large for $\Omega(N^2)$ inference algorithms such as variational MMSB [1] and the Gibbs sampling MMSB inference algorithm we developed for our experiments.

## 2 Triangular Motif Representation of a Network

In this work, we consider *undirected* networks over $N$ vertices, such as social networks. Most of the ideas presented here also generalize to directed networks, though the analysis is more involved since directed networks can generate more motifs than undirected ones. To prevent confusion, we shall use the term "1-edge" to refer to edges that exist between two vertices, and the term "0-edge" to refer to missing edges. Now, define a triangular motif $E_{ijk}$ involving vertices $i < j < k$ to be the type of subgraph over these 3 vertices. There are 4 basic classes of triangular motifs (Figure 1), distinguished by their number of 1-edges: full-triangle $\Delta_3$ (three 1-edges), 2-triangle $\Delta_2$ (two 1-edges), 1-triangle $\Delta_1$ (one 1-edge), and empty-triangle $\Delta_0$ (no 1-edges). The total number of triangles, over all 4 classes, is $\Theta(N^3)$. However, our goal is not to account for all 4 classes; instead, we will focus on $\Delta_3$ and $\Delta_2$ while ignoring $\Delta_1$ and $\Delta_0$. We have three primary motivations for this:

1. In the network literature, the most commonly studied "network motifs" [13], defined as patterns of significantly recurring inter-connections in complex networks, are the three-node *connected* subgraphs (namely $\Delta_3$ and $\Delta_2$) [13, 19, 9, 10, 16].

2. Since the full-triangle and 2-triangle classes are regarded as the basic structural elements of most networks [19, 13, 9, 10, 16], we naturally expect them to characterize most of the community structure in networks (cf. *network clustering coefficient*, as explained in the introduction). In particular, the $\Delta_3$ and $\Delta_2$ triangular motifs preserve almost all 1-edges from the original network: every 1-edge appears in some triangular motif $\Delta_2, \Delta_3$, except for isolated 1-edges (i.e. connected components of size 2), which are less interesting from a large-scale community detection perspective.

3. For real networks, which have far more 0- than 1-edges, focusing only on $\Delta_3$ and $\Delta_2$ greatly reduces the number of triangular motifs, via the following lemma:

**Lemma 1.** *The total number of $\Delta_3$'s and $\Delta_2$'s is upper bounded by $\sum_i \frac{1}{2}(D_i)(D_i - 1) = \Theta(\sum_i D_i^2)$, where $D_i$ is the degree of vertex $i$.*

*Proof.* Let $\mathcal{N}_i$ be the neighbor set of vertex $i$. For each vertex $i$, form the set $\mathcal{T}_i$ of tuples $(i, j, k)$ where $j < k$ and $j, k \in \mathcal{N}_i$, which represents the set of all pairs of neighbors of $i$. Because $j$ and $k$ are neighbors of $i$, for every tuple $(i, j, k) \in \mathcal{T}_i$, $E_{ijk}$ is either a $\Delta_3$ or a $\Delta_2$. It is easy to see that each $\Delta_2$ is accounted for by exactly one $\mathcal{T}_i$, where $i$ is the center vertex of the $\Delta_2$, and that each $\Delta_3$ is accounted for by three sets $\mathcal{T}_i, \mathcal{T}_j$ and $\mathcal{T}_k$, one for each vertex in the full-triangle. Thus, $\sum_i |\mathcal{T}_i| = \sum_i \frac{1}{2}(D_i)(D_i - 1)$ is an upper bound of the total number of $\Delta_3$'s and $\Delta_2$'s. $\qquad\square$

For networks with low maximum degree $\mathbf{D}$, $\Theta(\sum_i D_i^2) = \Theta(N\mathbf{D}^2)$ is typically much smaller than $\Theta(N^2)$, allowing triangular models to scale to larger networks than edge-based models. As for networks with high maximum degree, we suggest the following node-centric subsampling procedure, which we call $\delta$-subsampling: for each vertex $i$ with degree $D_i > \delta$ for some threshold $\delta$, sample $\frac{1}{2}\delta(\delta - 1)$ triangles without replacement and uniformly at random from $\mathcal{T}_i$; intuitively, this is similar to capping the network's maximum degree at $\mathbf{D}_s = \delta$. A full-triangle $\Delta_3$ associated with vertices $i, j$ and $k$ shall appear in the final subsample only if it has been subsampled from at least one of $\mathcal{T}_i, \mathcal{T}_j$ and $\mathcal{T}_k$. To obtain the set of all subsampled triangles $\Delta_2$ and $\Delta_3$, we simply take the union of subsampled triangles from each $\mathcal{T}_i$, discarding those full-triangles duplicated in the subsamples.

Although this node-centric subsampling does not preserve all properties of a network, such as the distribution of node degrees, it approximately preserves the local cluster properties of each vertex, thus capturing most of the community structure in networks. Specifically, the "local" clustering coefficient (LCC) of each vertex $i$, defined as the ratio of $\#(\Delta_3)$ touching $i$ to $\#(\Delta_3, \Delta_2)$ touching $i$, is well-preserved. This follows from subsampling the $\Delta_3$ and $\Delta_2$'s at $i$ uniformly at random, though the LCC has a small upwards bias since each $\Delta_3$ may also be sampled by the other two vertices $j$ and $k$. Hence, we expect community detection based on the subsampled triangles to be nearly as accurate as with the original set of triangles — which our experiments will show.

We note that other subsampling strategies [11, 22] preserve various network properties, such as degree distribution, diameter, and inter-node random walk times. In our triangular model, the main property of interest is the distribution over $\Delta_3$ and $\Delta_2$, analogous to how latent factor models and MMSB model distributions over 0- and 1-edges. Thus, subsampling strategies that preserve $\Delta_3/\Delta_2$ distributions (e.g. our $\delta$-subsampling) would be appropriate for our model. In contrast, 0/1-edge subsampling for MMSB and latent factor models is difficult: most networks have $\Theta(N^2)$ 0-edges but only $o(N^2)$ 1-edges, thus sampling $o(N^2)$ 0/1-edges leads to high variance in their distribution.

## 3 Mixed-Membership Triangular Model

Given a network, now represented by triangular motifs $\Delta_3$ and $\Delta_2$, our goal is to perform community detection for each network vertex $i$, in the same sense as what an MMSB model would enable. Under an MMSB, each vertex $i$ is assigned to a mixture over communities, as opposed to traditional single-membership community detection, which assigns each vertex to exactly one community. By taking a mixed-membership approach, one gains many benefits over single-membership models, such as outlier detection, improved visualization, and better interpretability [2, 1].

Following a design principle similar to the one underlying the MMSB models, we now present a new mixed-membership network model built on the more parsimonious triangular representation. For each triplet of vertices $i, j, k \in \{1, \ldots, N\}$, $i < j < k$, if the subgraph on $i, j, k$ is a 2-triangle with $i$, $j$, or $k$ at the center, then let $E_{ijk} = 1$, 2 or 3 respectively, and if the subgraph is a full-triangle, then let $E_{ijk} = 4$. Whenever $i, j, k$ corresponds to a 1- or an empty-triangle, we do not model $E_{ijk}$. We assume $K$ latent communities, and that each vertex takes a distribution (i.e. mixed-membership) over them. The observed bag-of-triangles $\{E_{ijk}\}$ is generated according to (1) the distribution over community-memberships at each vertex, and (2) a tensor of triangle generation probabilities, containing different triangle probabilities for different combinations of communities.

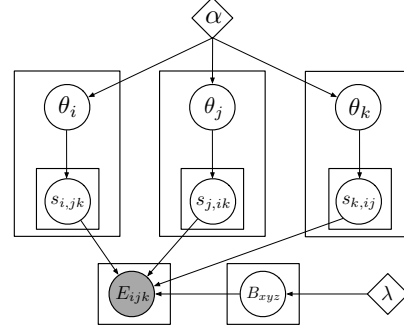

Figure 2: Graphical model representation for MMTM, our mixed-membership model over triangular motifs.

More specifically, each vertex $i$ is associated with a community mixed-membership vector $\theta_i \in \Delta^{K-1}$ restricted to the $(K-1)$-simplex $\Delta^{K-1}$. This mixed-membership vector $\theta_i$ is used to generate community indicators $s_{i,jk} \in \{1, \ldots, K\}$, each of which represents the community chosen by vertex $i$ when it is forming a triangle with vertices $j$ and $k$. The probability of observing a triangular motif $E_{ijk}$ depends on the community-triplet $s_{i,jk}, s_{j,ik}, s_{k,ij}$, and a tensor of multinomial parameters $B$. Let $x, y, z \in \{1, \ldots, K\}$ be the values of $s_{i,jk}, s_{j,ik}, s_{k,ij}$, and assume WLOG that $x < y < z$[1]. Then, $B_{xyz} \in \Delta^2$ represents the probabilities of generating the 4 triangular motifs[2] among vertices $i, j$ and $k$. In detail, $B_{xyz,1}$ is the probability of the 2-triangle whose center vertex has community $x$, and analogously for $B_{xyz,2}$ and community $y$, and for $B_{xyz,3}$ and community $z$; $B_{xyz,4}$ is the probability of the full-triangle.

The MMTM generative model is summarized below; see Figure 2 for a graphical model illustration.

- Triangle tensor $B_{xyz} \sim \text{Dirichlet}(\lambda)$ for all $x, y, z \in \{1, \ldots, K\}$, where $x < y < z$
- Community mixed-membership vectors $\theta_i \sim \text{Dirichlet}(\alpha)$ for all $i \in \{1, \ldots, N\}$
- For each triplet $(i, j, k)$ where $i < j < k$,
  - Community indices $s_{i,jk} \sim \text{Discrete}(\theta_i)$, $s_{j,ik} \sim \text{Discrete}(\theta_j)$, $s_{k,ij} \sim \text{Discrete}(\theta_k)$.
  - Generate the triangular motif $E_{ijk}$ based on $B_{xyz}$ and the ordered values of $s_{i,jk}, s_{j,ik}, s_{k,ij}$; see Table 1 for the exact conditional probabilities. There are 6 entries in Table 1, corresponding to the 6 possible orderings of $s_{i,jk}, s_{j,ik}, s_{k,ij}$.

## 4 Inference

We adopt a collapsed, blocked Gibbs sampling approach, where $\theta$ and $B$ have been integrated out. Thus, only the community indices $\mathbf{s}$ need to be sampled. For each triplet $(i, j, k)$ where $i < j < k$,

$$\mathbb{P}\left(s_{i,jk}, s_{j,ik}, s_{k,ij} \mid \mathbf{s}_{-ijk}, \mathbf{E}, \alpha, \lambda\right) \propto \mathbb{P}\left(E_{ijk} | \mathbf{E}_{-ijk}, \mathbf{s}, \lambda\right) \mathbb{P}\left(s_{i,jk} \mid \mathbf{s}_{i,-jk}, \alpha\right)$$
$$\mathbb{P}\left(s_{j,ik} \mid \mathbf{s}_{j,-ik}, \alpha\right) \mathbb{P}\left(s_{k,ij} \mid \mathbf{s}_{k,-ij}, \alpha\right),$$

| Order | Conditional probability of $E_{ijk} \in \{1,2,3,4\}$ |
|---|---|
| $s_{i,jk} < s_{j,ik} < s_{k,ij}$ | Discrete($[B_{xyz,1}, B_{xyz,2}, B_{xyz,3}, B_{xyz,4}]$) |
| $s_{i,jk} < s_{k,ij} < s_{j,ik}$ | Discrete($[B_{xyz,1}, B_{xyz,3}, B_{xyz,2}, B_{xyz,4}]$) |
| $s_{j,ik} < s_{i,jk} < s_{k,ij}$ | Discrete($[B_{xyz,2}, B_{xyz,1}, B_{xyz,3}, B_{xyz,4}]$) |
| $s_{j,ik} < s_{k,ij} < s_{i,jk}$ | Discrete($[B_{xyz,3}, B_{xyz,1}, B_{xyz,2}, B_{xyz,4}]$) |
| $s_{k,ij} < s_{i,jk} < s_{j,ik}$ | Discrete($[B_{xyz,2}, B_{xyz,3}, B_{xyz,1}, B_{xyz,4}]$) |
| $s_{k,ij} < s_{j,ik} < s_{i,jk}$ | Discrete($[B_{xyz,3}, B_{xyz,2}, B_{xyz,1}, B_{xyz,4}]$) |

Table 1: Conditional probabilities of $E_{ijk}$ given $s_{i,jk}, s_{j,ik}$ and $s_{k,ij}$. We define $x, y, z$ to be the ordered (i.e. sorted) values of $s_{i,jk}, s_{j,ik}, s_{k,ij}$.

where $\mathbf{s}_{-ijk}$ is the set of all community memberships except for $s_{i,jk}, s_{j,ik}, s_{k,ij}$, and $\mathbf{s}_{i,-jk}$ is the set of all community memberships of vertex i except for $s_{i,jk}$. The last three terms are predictive distributions of a multinomial-Dirichlet model, with the multinomial parameter $\theta$ marginalized out:

$$\mathbb{P}\left(s_{i,jk} \mid \mathbf{s}_{i,-jk}, \alpha\right) = \frac{\#\left[\mathbf{s}_{i,-jk} = s_{i,jk}\right] + \alpha}{\#\left[\mathbf{s}_{i,-jk}\right] + K\alpha}.$$

The first term is also a multinomial-Dirichlet predictive distribution (refer to appendix for details).

# 5 Comparing Mixed-Membership Network Models on Synthetic Networks

For a mixed-membership network model to be useful, it must recover some meaningful notion of mixed community membership for each vertex. The precise definition of network community has been a subject of much debate, and various notions of community [1, 15, 17, 12, 6] have been proposed under different motivations. Our MMTM, too, conveys another notion of community based on membership in full triangles $\Delta_3$ and 2-triangles $\Delta_2$, which are key aspects of network clustering coefficients. In our simulations, we shall compare our MMTM against an adjacency-matrix-based model (MMSB), in terms of how well they recover mixed-memberships from networks generated under a range of assumptions. Note that some of these synthetic networks will not match the generative assumptions of either our model or MMSB; this is intentional, as we want to compare the performance of both models under model misspecification.

We shall also demonstrate that MMTM leads to faster inference, particularly when $\delta$-subsampling triangles (as described in Section 2). Intuitively, we expect the mixed-membership recovery of our inference algorithm to depend on (a) the degree distribution of the network, and (b) the "degree limit" $\delta$ used in subsampling the network; performance should increase as the number of vertices $i$ having degree $D_i \leq \delta$ goes up. In particular, our experiments will demonstrate that subsampling yields good performance even when the network contains a few vertices with very large degree $D_i$ (a characteristic of many real-world networks).

**Synthetic networks** We compared our MMTM to MMSB[3] [1] on multiple synthetic networks, evaluating them according to how well their inference algorithms recovered the vertex mixed-membership vectors $\theta_i$. Each network was generated from $N = 4,000$ mixed-membership vectors $\theta_i$ of dimensionality $K = 5$ (i.e. 5 possible communities), according to one of several models:

1. The **Mixed Membership Stochastic Blockmodel** [1], an admixture generalization of the stochastic blockmodel. The probability of a link from $i$ to $j$ is $\theta_i B \theta_j$ for some block matrix $B$, and we convert all directed edges into undirected edges. In our experiments, we use a $B$ with on-diagonal elements $B_{aa} = 1/80$, and off-diagonal elements $B_{ab} = 1/800$. Our values of $B$ are lower than typically seen in the literature, because they are intended to replicate the 1-edge density of real-world networks with size around $N = 4,000$.

2. A simplex **Latent position model**, where the probability of a link between $i, j$ is $\gamma(1 - \frac{1}{2}||\theta_i - \theta_j||_1)$ for some scaling parameter $\gamma$. In other words, the closer that $\theta_i$ and $\theta_j$ are, the higher the link probability. Note that $0 \leq ||\theta_i - \theta_j||_1 \leq 2$, because $\theta_i$ and $\theta_j$ lie in the simplex. We choose $\gamma = 1/40$, again to reproduce the 1-edge density seen in real networks.

3. A **"Biased" scale-free model** that combines the preferred attachment model [7] with a mixed-membership model. Specifically, we generated $M = 60,000$ 1-edges as follows: (a) pick a vertex $i$ with probability proportional to its degree; (b) randomly pick a destination community $k$ from $\theta_i$; (c) find the set $V_k$ of all vertices $v$ such that $\theta_{vk}$ is the largest element of $\theta_v$ (i.e. the vertices that mostly belong to community $k$); (d) within $V_k$, pick the destination vertex $j$ with probability proportional to its degree. The resulting network

| | #0,1-edges | #1-edges | max($D_i$) | #$\Delta_3, \Delta_2$ | $\delta = 20$ | $\delta = 15$ | $\delta = 10$ | $\delta = 5$ |
|---|---|---|---|---|---|---|---|---|
| MMSB | 7,998,000 | 55,696 | 51 | 1,541,085 | 749,018 | 418,764 | 179,841 | 39,996 |
| Latent position | ‖ | 56,077 | 51 | 1,562,710 | 746,979 | 418,448 | 179,757 | 39,988 |
| Biased scale-free | ‖ | 60,000 | 231 | 3,176,927 | 497,737 | 304,866 | 144,206 | 35,470 |
| Pure membership | ‖ | 55,651 | 44 | 1,533,365 | 746,796 | 418,222 | 179,693 | 39,986 |

Table 2: Number of edges, maximum degree, and number of 3- and 2-edge triangles $\Delta_3, \Delta_2$ for each $N = 4,000$ synthetic network, as well as #triangles when subsampling at various degree thresholds $\delta$. MMSB inference is linear in #0,1-edges, while our MMTM's inference is linear in #$\Delta_3, \Delta_2$.

exhibits both a block diagonal structure, as well as a power-law degree distribution. In contrast, the other two models have binomial (i.e. Gaussian-like) degree distributions.

To use these models, we must input mixed-memberships $\theta_i$. These were generated as follows:

1. Divide the $N = 4,000$ vertices into 5 groups of size 800. Assign each group to a (different) dominant community $k \in \{1, \ldots, 5\}$.
2. Within each group:
   (a) Pick 160 vertices to have mixed-membership in 3 communities: 0.8 in the dominant community $k$, and 0.1 in two other randomly chosen communities.
   (b) The remaining 640 vertices have mixed-membership in 2 communities: 0.8 in the dominant community $k$, and 0.2 in one other randomly chosen community.

In other words, every vertex has a dominant community, and one or two other minor communities. Using these $\theta_i$'s, we generated one synthetic network for each of the three models described. In addition, we generated a fourth "pure membership" network under the MMSB model, using pure $\theta_i$'s with full membership in the dominant community. This network represents the special case of single-community membership. Statistics for all 4 networks can be found in Table 2.

**Inference and Evaluation** For our MMTM[4], we used our collapsed, blocked Gibbs sampler for inference. The hyperparameters were fixed at $\alpha, \lambda = 0.1$ and $K = 5$, and we ran each experiment for 2,000 iterations. For evaluation, we estimated all $\theta_i$'s using the last sample, and scored the estimates according to $\sum_i ||\hat{\theta}_i - \theta_i||_2$, the sum of $\ell_2$ distances of each estimate $\hat{\theta}_i$ from its true value $\theta_i$. These results were taken under the *most favorable permutation* for the $\hat{\theta}_i$'s, in order to avoid the permutation non-identifiability issue. We repeated every experiment 5 times.

To investigate the effect of $\delta$-subsampling triangles (Section 2), we repeated every MMTM experiment under four different values of $\delta$: 20, 15, 10 and 5. The triangles were subsampled prior to running the Gibbs sampler, and they remained fixed during inference.

With MMSB, we opted not to use the variational inference algorithm of [1], because we wanted our experiments to be, as far as possible, a comparison of models rather than inference techniques. To accomplish this, we derived a collapsed, blocked Gibbs sampler for the MMSB model, with added Beta hyperparameters $\lambda_1, \lambda_2$ on each element of the block matrix $B$. The mixed-membership vectors $\theta_i$ ($\pi_i$ in the original paper) and blockmatrix $B$ were integrated out, and we Gibbs sampled each edge $(i, j)$'s associated community indicators $z_{i \rightarrow j}, z_{i \leftarrow j}$ in a block fashion. Hence, this MMSB sampler uses the exact same techniques as our MMTM sampler, ensuring that we are comparing models rather than inference strategies. Furthermore, its per-iteration runtime is still $\Theta(N^2)$, equal to the original MMSB variational algorithm. All experiments were conducted in exactly the same manner as with MMTM, with the MMSB hyperparameters fixed at $\alpha, \lambda_1, \lambda_2 = 0.1$ and $K = 5$.

**Results** Figure 3 plots the cumulative $\ell_2$ error for each experiment, as well as the time taken per trial. On all 4 networks, the full MMTM model performs better than MMSB — even on the MMSB-generated network! MMTM also requires less runtime for all but the biased scale-free network, which has a much larger maximum degree than the others (Table 2). Furthermore, $\delta$-subsampling is effective: MMTM with $\delta = 20$ runs faster than full MMTM, and still outperforms MMSB while approaching full MMTM in accuracy. The runtime benefit is most noticeable on the biased scale-free network, underscoring the need to subsample real-world networks with high maximum degree.

We hypothesize MMSB's poorer performance on networks of this size ($N = 4,000$) results from having $\Theta(N^2)$ latent variables, while noting that the literature has only considered smaller $N < 1,000$ networks [1]. Compared to MMTM, having many latent variables not only increases runtime per iteration, but also the number of iterations required for convergence, since the latent variable state space grows exponentially with the number of latent variables. In support of this, we have observed

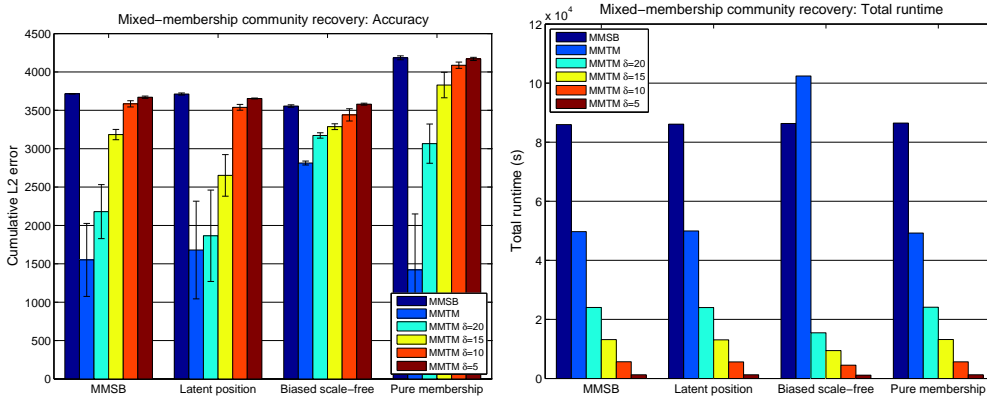

Figure 3: Mixed-membership community recovery task: Cumulative $\ell_2$ errors and runtime per trial for MMSB, MMTM and MMTM with $\delta$-subsampling, on $N = 4,000$ synthetic networks.

that the MMSB sampler's complete log-likelihood fluctuates greatly across all 2000 iterations; in contrast, the MMTM sampler plateaus within 500 iterations, and remains stable.

**Scalability Experiments**    Although the preceding $N = 4,000$ experiments appear fairly small, in actual fact, they are close to the feasible limit for adjacency-matrix-based models like MMSB. To demonstrate this, we generated four networks with sizes $N \in \{1000, 4000, 10000, 40000\}$ from the MMSB generative model. The generative parameters for the $N = 4,000$ network are identical to our earlier experiment, while the parameters for the other three network sizes were adjusted to maintain the same average degree[5]. We then ran the MMSB, MMTM, and MMTM with $\delta$-subsampling inference algorithms on all 4 networks, and plotted the average per-iteration runtime in Figure 4.

The figure clearly exposes the scalability differences between MMSB and MMTM. The $\delta$-subsampled MMTM experiments show linear runtime dependence on $N$, which is expected since the number of subsampled triangles is $\mathcal{O}(N\delta^2)$. The full MMTM experiment is also roughly linear — though we caution that this is not necessarily true for all networks, particularly high maximum degree ones such as scale-free networks. Conversely, MMSB shows a clear quadratic dependence on $N$. In fact, we had to omit the MMSB $N = 40,000$ experiment because the latent variables would not fit in memory, and even if they did, the extrapolated runtime would have been unreasonably long.

## 6    A Larger Network Demonstration

The MMTM model with $\delta$-subsampling scales to even larger networks than the ones we have been discussing. To demonstrate this, we ran the MMTM Gibbs sampler with $\delta = 20$ on the SNAP Stanford Web Graph[6], containing $N = 281,903$ vertices (webpages), $2,312,497$ 1-edges, and approximately 4 billiion 2- and 3-edge triangles $\Delta_3, \Delta_2$, which we reduced to $11,353,778$ via $\delta = 20$-subsampling. Note that the vast majority of triangles are associated with exceptionally high-degree vertices, which make up a small fraction of the network. By using $\delta$-subsampling, we limited the number of triangles that come from such vertices, thus making the network feasible for MMTM. We ran the MMTM sampler with settings identical to our synthetic experiments: 2,000 sampling iterations, hyperparameters fixed to $\alpha, \lambda = 0.1$. The experiment took 74 hours, and we observed log-likelihood convergence within 500 iterations.

The recovered mixed-membership vectors $\theta_i$ are visualized in Figure 5. A key challenge is that the $\theta_i$ exist in the 4-simplex $\Delta^4$, which is difficult to visualize in two dimensions. To overcome this, Figure 5 uses both position and color to communicate the values of $\theta_i$. Every vertex $i$ is displayed as a circle $c_i$, whose size is proportional to the network degree of $i$. The position of $c_i$ is equal to a convex combination of the 5 pentagon corners' $(x, y)$ coordinates, where the coordinates are weighted by the elements of $\theta_i$. In particular, circles $c_i$ at the pentagon's corners represent single-membership $\theta_i$'s, while circles on the lines connecting the corners represent $\theta_i$'s with mixed-membership in 2 communities. All other circles represent $\theta_i$'s with mixed-membership in $\geq 3$ communities. Furthermore, each circle $c_i$'s color is also a $\theta_i$-weighted convex combination, this time of the RGB values of 5 colors: blue, green, red, cyan and purple. This use of color helps distinguish between vertices with 2 versus 3 or more communities: for example, even though the largest circle sits on the blue-red line (which initially suggests mixed-membership in 2 communities), its dark green color actually comes from mixed-membership in 3 communities: green, red and cyan.

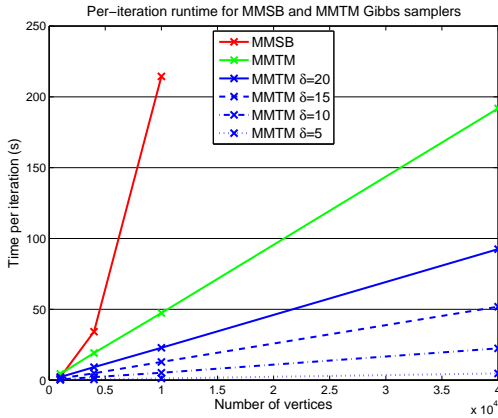

Figure 4: Per-iteration runtimes for MMSB, MMTM and MMTM with $\delta$-subsampling, on synthetic networks with $N$ ranging from 1,000 to 40,000, but with constant average degree.

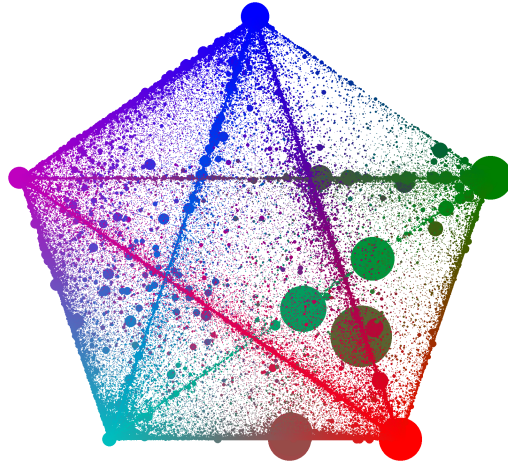

Figure 5: $N = 281,903$ Stanford web graph, MMTM mixed-membership visualization.

Most high-degree vertices (large circles) are found at the pentagon's corners, leading to the intuitive conclusion that the five communities are centered on hub webpages with many links. Interestingly, the highest-degree vertices are all mixed-membership, suggesting that these webpages (which are mostly frontpages) lie on the boundaries between the communities. Finally, if we focus on the sets of vertices near each corner, we see that the green and red sets have distinct degree (i.e. circle size) distributions, suggesting that those communities may be functionally different from the other three.

## 7 Future Work and Conclusion

We have focused exclusively on triangular motifs because of their popularity in the literature, their relationship to community structure through the network clustering coefficient, and the ability to subsample them in a natural, node-centric fashion with minor impact on accuracy. However, the bag-of-network-motifs idea extends beyond triangles — one could easily consider subgraphs over 4 or more vertices, as in [13]. As with triangular motifs, it is algorithmically infeasible to consider all possible subgraphs; rather, we must focus our attention on a meaningful subset of them. Nevertheless, higher order motifs could be more suited for particular tasks, thus meriting their investigation.

In modeling terms, we have applied triangular motifs to a generative mixed-membership setting, which is suitable for visualization but not necessarily for attribute prediction. Recent developments in constrained learning of generative models [23, 24] have yielded significant improvements in predictive accuracy, and these techniques are also applicable to mixed-membership triangular modeling. Also, given how well $\delta = 20$-subsampling works for MMTM at $N = 4,000$, the next step would be investigating how to adaptively choose $\delta$ as $N$ increases, in order to achieve good performance.

To summarize, we have introduced the bag-of-triangles representation as a parsimonius alternative to the network adjacency matrix, and developed a model (MMTM) and inference algorithm for mixed-membership community detection in networks. Compared to mixed-membership models that use the adjacency matrix (exemplified by MMSB), our model features a much smaller latent variable space, leading to faster inference and better performance at mixed-membership recovery. When combined with triangle subsampling, our model and inference algorithm scale easily to networks with 100,000s of vertices, which are completely infeasible for $\Theta(N^2)$ adjacency-matrix-based models — the adjacency matrix might not even fit in memory, to say nothing of runtime.

As a final note, we speculate that the local nature of the triangles lends itself better to parallel inference than the adjacency matrix representation; it may be possible to find good "triangle separators", small subsets of triangles that divide the remaining triangles into large, non-vertex-overlapping subsets, which can then be inferred in parallel. This is similar to classical 1-edge separators that divide networks into non-overlapping subgraphs, which are unfortunately inapplicable to adjacency-matrix-based models, as they require separators over both the 0- and 1-edges. With triangle separators, we expect triangle models to scale to networks with millions of vertices and more.

**Acknowledgments**
This work was supported by AFOSR FA9550010247, NIH 1R01GM093156 to Eric P. Xing. Qirong Ho is supported by an Agency for Science, Research and Technology, Singapore fellowship. Junming Yin is a Lane Fellow under the Ray and Stephanie Lane Center for Computational Biology.

## Footnotes

[1]The cases $x = y = z, x = y < z$ and $x < y = z$ require special treatment, due to ambiguity cased by having identical communities. In the interest of keeping our discussion at a high level, we shall refer the reader to the appendix for these cases.

[2]It is possible to generate a set of triangles that does not correspond to a network, e.g. a 2-triangle centered on $i$ for $(i, j, k)$ followed by a 3-triangle for $(j, k, \ell)$, which produces a mismatch on the edge $(j, k)$. This is a consequence of using a bag-of-triangles model, just as the bag-of-words model in Latent Dirichlet Allocation can generate sets of words that do not correspond to grammatical sentences. In practice, this is not an issue for either our model or LDA, as both models are used for mixed-membership recovery, rather than data simulation.

[3]MMSB is applicable to both directed and undirected networks; our experiments use the latter.

[4]As explained in Section 2, we first need to preprocess the network adjacency list into the $\Delta_3, \Delta_2$ triangle representation. The time required is linear in the number of $\Delta_3, \Delta_2$ triangles, and is insignificant compared to the actual cost of MMTM inference.

[5]Note that the maximum degree still increases with $N$, because MMSB has a binomial degree distribution.

[6]Available at http://snap.stanford.edu/data/web-Stanford.html

# References

[1] E.M. Airoldi, D.M. Blei, S.E. Fienberg, and E.P. Xing. Mixed membership stochastic blockmodels. *The Journal of Machine Learning Research*, 9:1981–2014, 2008.

[2] D.M. Blei, A.Y. Ng, and M.I. Jordan. Latent dirichlet allocation. *The Journal of Machine Learning Research*, 3:993–1022, 2003.

[3] L. Cao and L. Fei-Fei. Spatially coherent latent topic model for concurrent segmentation and classification of objects and scenes. In *ICCV 2007*, pages 1–8. IEEE, 2007.

[4] B. Fulkerson, A. Vedaldi, and S. Soatto. Class segmentation and object localization with superpixel neighborhoods. In *ICCV 2009*, pages 670–677. IEEE, 2009.

[5] Q. Ho, J. Eisenstein, and E.P. Xing. Document hierarchies from text and links. In *Proceedings of the 21st international conference on World Wide Web*, pages 739–748. ACM, 2012.

[6] Q. Ho, A. Parikh, L. Song, and EP Xing. Multiscale community blockmodel for network exploration. In *Proceedings of the 14th International Conference on Artificial Intelligence and Statistics*, 2011.

[7] M.J. Keeling and K.T.D. Eames. Networks and epidemic models. *Journal of the Royal Society Interface*, 2(4):295–307, 2005.

[8] R. Kondor, N. Shervashidze, and K.M. Borgwardt. The graphlet spectrum. In *Proceedings of the 26th Annual International Conference on Machine Learning*, pages 529–536. ACM, 2009.

[9] D. Krackhardt and M. Handcock. Heider vs simmel: Emergent features in dynamic structures. *Statistical Network Analysis: Models, Issues, and New Directions*, pages 14–27, 2007.

[10] J. Leskovec, L. Backstrom, R. Kumar, and A. Tomkins. Microscopic evolution of social networks. In *Proceeding of the 14th ACM SIGKDD international conference on Knowledge discovery and data mining*, pages 462–470. ACM, 2008.

[11] J. Leskovec and C. Faloutsos. Sampling from large graphs. In *Proceedings of the 12th ACM SIGKDD international conference on Knowledge discovery and data mining*, pages 631–636. ACM, 2006.

[12] K.T. Miller, T.L. Griffiths, and M.I. Jordan. Nonparametric latent feature models for link prediction. *Advances in Neural Information Processing Systems (NIPS)*, pages 1276–1284, 2009.

[13] R. Milo, S. Shen-Orr, S. Itzkovitz, N. Kashtan, D. Chklovskii, and U. Alon. Network motifs: Simple building blocks of complex networks. *Science*, 298(5594):824–827, 2002.

[14] R.M. Nallapati, A. Ahmed, E.P. Xing, and W.W. Cohen. Joint latent topic models for text and citations. In *Proceeding of the 14th ACM SIGKDD international conference on Knowledge discovery and data mining*, pages 542–550. ACM, 2008.

[15] M.E.J. Newman. Modularity and community structure in networks. *Proceedings of the National Academy of Sciences*, 103(23):8577–8582, 2006.

[16] M.E.J. Newman and J. Park. Why social networks are different from other types of networks. *Arxiv preprint cond-mat/0305612*, 2003.

[17] A.Y. Ng, M.I. Jordan, and Y. Weiss. On spectral clustering: Analysis and an algorithm. *Advances in neural information processing systems*, 2:849–856, 2002.

[18] N. Shervashidze, SVN Vishwanathan, T. Petri, K. Mehlhorn, and K. Borgwardt. Efficient graphlet kernels for large graph comparison. In *Proceedings of the International Workshop on Artificial Intelligence and Statistics. Society for Artificial Intelligence and Statistics*, 2009.

[19] G. Simmel and K.H. Wolff. *The Sociology of Georg Simmel*. Free Press, 1950.

[20] T.A.B. Snijders. Markov chain monte carlo estimation of exponential random graph models. *Journal of Social Structure*, 3(2):1–40, 2002.

[21] C.E. Tsourakakis. Fast counting of triangles in large real networks without counting: Algorithms and laws. In *Data Mining, 2008. ICDM'08. Eighth IEEE International Conference on*, pages 608–617. IEEE, 2008.

[22] A. Vattani, D. Chakrabarti, and M. Gurevich. Preserving personalized pagerank in subgraphs. In *ICML 2011*, 2011.

[23] J. Zhu, A. Ahmed, and E.P. Xing. Medlda: maximum margin supervised topic models for regression and classification. In *Proceedings of the 26th Annual International Conference on Machine Learning*, pages 1257–1264. ACM, 2009.

[24] J. Zhu, N. Chen, and E.P. Xing. Infinite latent svm for classification and multi-task learning. *Advances in Neural Information Processing Systems*, 25.

